# Two view learning: SVM-2K, Theory and Practice

**Jason D.R. Farquhar**
jdrf99r@ecs.soton.ac.uk

**David R. Hardoon**
drh@ecs.soton.ac.uk

**Hongying Meng**
hongying@cs.york.ac.uk

**John Shawe-Taylor**
jst@ecs.soton.ac.uk

**Sandor Szedmak**
ss03v@ecs.soton.ac.uk

School of Electronics and Computer Science,
University of Southampton, Southampton, England

## Abstract

Kernel methods make it relatively easy to define complex high-dimensional feature spaces. This raises the question of how we can identify the relevant subspaces for a particular learning task. When two views of the same phenomenon are available kernel Canonical Correlation Analysis (KCCA) has been shown to be an effective preprocessing step that can improve the performance of classification algorithms such as the Support Vector Machine (SVM). This paper takes this observation to its logical conclusion and proposes a method that combines this two stage learning (KCCA followed by SVM) into a single optimisation termed SVM-2K. We present both experimental and theoretical analysis of the approach showing encouraging results and insights.

## 1   Introduction

Kernel methods enable us to work with high dimensional feature spaces by defining weight vectors implicitly as linear combinations of the training examples. This even makes it practical to learn in infinite dimensional spaces as for example when using the Gaussian kernel. The Gaussian kernel is an extreme example, but techniques have been developed to define kernels for a range of different datatypes, in many cases characterised by very high dimensionality. Examples are the string kernels for text, graph kernels for graphs, marginal kernels, kernels for image data, etc.

With this plethora of high dimensional representations it is frequently helpful to assist learning algorithms by preprocessing the feature space in projecting the data into a low dimensional subspace that contains the relevant information for the learning task. Methods of performing this include principle components analysis (PCA) [7], partial least squares [8], kernel independent component analysis (KICA) [1] and kernel canonical correlation analysis (KCCA) [5].

The last method requires two views of the data both of which contain all of the relevant information for the learning task, but which individually contain representation specific details that are different and irrelevant. Perhaps the simplest example of this situation is a paired document corpus in which we have the same information in two languages. KCCA attempts to isolate feature space directions that correlate between the two views and hence might be expected to represent the common relevant information. Hence, one can view this preprocessing as a denoising of the individual representations through cross-correlating them.

Experiments have shown how using this as a preprocessing step can improve subsequent analysis in for example classification experiments using a support vector machine (SVM) [6]. This is explained by the fact that the signal to noise ratio has improved in the identified subspace.

Though the combination of KCCA and SVM seems effective, there appears no guarantee that the directions identified by KCCA will be best suited to the classification task. This paper therefore looks at the possibility of combining the two distinct stages of KCCA and SVM into a single optimisation that will be termed SVM-2K.

The next section introduces the new algorithm and discusses its structure. Experiments are then given showing the performance of the algorithm on an image classification task.

Though the performance is encouraging it is in many ways counter-intuitive, leading to speculation about why an improvement is seen. To investigate this question an analysis of its generalisation properties is given in the following two sections, before drawing conclusions.

## 2  SVM-2K Algorithm

We assume that we are given two views of the same data, one expressed through a feature projection $\phi_A$ with corresponding kernel $\kappa_A$ and the other through a feature projection $\phi_B$ with kernel $\kappa_B$. A paired data set is then given by a set

$$S = \{(\phi_A(\mathbf{x}_1), \phi_B(\mathbf{x}_1)), \ldots, (\phi_A(\mathbf{x}_\ell), \phi_B(\mathbf{x}_\ell))\},$$

where for example $\phi_A$ could be the feature vector associated with one language and $\phi_B$ that associated with a second language. For a classification task each data item would also include a label.

The KCCA algorithm looks for directions in the two feature spaces such that when the training data is projected onto those directions the two vectors (one for each view) of values obtained are maximally correlated. One can also characterise these directions as those that minimise the two norm between the two vectors under the constraint that they both have norm 1 [5].

We can think of this as constraining the choice of weight vectors in the two spaces. KCCA would typically find a sequence of projection directions of dimension anywhere between 50 and 500 that can then be used as the feature space for training an SVM [6].

An SVM can be thought of as a 1-dimensional projection followed by thresholding, so SVM-2K combines the two steps by introducing the constraint of similarity between two 1-dimensional projections identifying two distinct SVMs one in each of the two feature spaces. The extra constraint is chosen slightly differently from the 2-norm that characterises KCCA. We rather take an $\epsilon$-insensitive 1-norm using slack variables to measure the amount by which points fail to meet $\epsilon$ similarity:

$$|\langle \mathbf{w}_A, \phi_A(\mathbf{x}_i)\rangle + b_A - \langle \mathbf{w}_B, \phi_B(\mathbf{x}_i)\rangle - b_B| \leq \eta_i + \epsilon,$$

where $\mathbf{w}_A$, $b_A$ ($\mathbf{w}_B$, $b_B$) are the weight and threshold of the first (second) SVM.

Combining this constraint with the usual 1-norm SVM constraints and allowing different

regularisation constants gives the following optimisation:

$$\min L = \frac{1}{2}\|\mathbf{w}_A\|^2 + \frac{1}{2}\|\mathbf{w}_B\|^2 + C^A\sum_{i=1}^{\ell}\xi_i^A + C^B\sum_{i=1}^{\ell}\xi_i^B + D\sum_{i=1}^{\ell}\eta_i$$

$$\text{such that} \quad |\langle\mathbf{w}_A,\phi_A(\mathbf{x}_i)\rangle + b_A - \langle\mathbf{w}_B,\phi_B(\mathbf{x}_i)\rangle - b_B| \le \eta_i + \epsilon$$
$$y_i\left(\langle\mathbf{w}_A,\phi_A(\mathbf{x}_i)\rangle + b_A\right) \ge 1 - \xi_i^A$$
$$y_i\left(\langle\mathbf{w}_B,\phi_B(\mathbf{x}_i)\rangle + b_B\right) \ge 1 - \xi_i^B$$
$$\xi_i^A \ge 0, \quad \xi_i^B \ge 0, \quad \eta_i \ge 0 \quad \text{all for} \quad 1 \le i \le \ell.$$

Let $\hat{\mathbf{w}}_A$, $\hat{\mathbf{w}}_B$, $\hat{b}_A$, $\hat{b}_B$ be the solution to this optimisation problem. The final SVM-2K decision function is then $h(x) = \text{sign}(f(x))$, where

$$f(x) = 0.5\left(\langle\hat{\mathbf{w}}_A,\phi_A(x)\rangle + \hat{b}_A + \langle\hat{\mathbf{w}}_B,\phi_B(x)\rangle + \hat{b}_B\right) = 0.5\left(f_A(x) + f_B(x)\right).$$

Applying the usual Lagrange multiplier techniques we arrive at the following dual problem:

$$\max W = -\frac{1}{2}\sum_{i,j=1}^{\ell}\left(g_i^A g_j^A \kappa_A(\mathbf{x}_i,\mathbf{x}_j) + g_i^B g_j^B \kappa_B(\mathbf{x}_i,\mathbf{x}_j)\right) + \sum_{i=1}^{\ell}(\alpha_i^A + \alpha_i^B)$$

$$\text{such that} \quad g_i^A = \alpha_i^A y_i - \beta_i^+ + \beta_i^-, \quad g_i^B = \alpha_i^B y_i + \beta_i^+ - \beta_i^-,$$
$$\sum_{i=1}^{\ell} g_i^A = 0 = \sum_{i=1}^{\ell} g_i^B,$$
$$0 \le \alpha_i^{A/B} \le C^{A/B}$$
$$0 \le \beta_i^{+/-}, \quad \beta_i^+ + \beta_i^- \le D$$

with the functions

$$f_{A/B}(x) = \sum_{i=1}^{\ell} g_i^{A/B} \kappa_{A/B}(\mathbf{x}_i,x) + b_{A/B}.$$

## 3  Experimental results

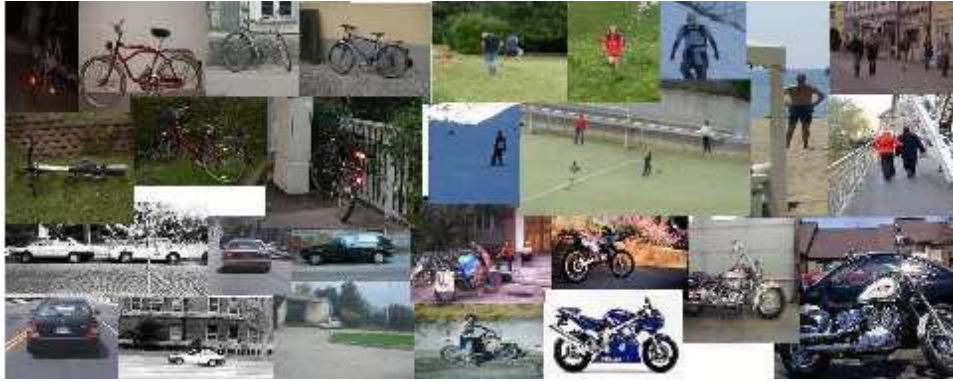

Figure 1: Typical example images from the PASCAL VOC challenge database. Classes are; Bikes (top-left), People (top-right), Cars (bottom-left) and Motorbikes (bottom-right).

The performance of the algorithms developed in this paper we evaluated on PASCAL Visual Object Classes (VOC) challenge dataset test1[1]. This is a new dataset consisting of four object classes in realistic scenes. The object classes are, motorbikes (M), bicycles (B), people (P) and cars (C) with the dataset containing 684 training set images consisting of (214, 114, 84, 272) images in each class and 689 test set images with (216, 114, 84, 275) for each class. As can be seen in Figure 1 this is a very challenging dataset with objects of widely varying type, pose, illumination, occlusion, background, etc.

The task is to classify the image according to whether it contains a given object type. We tested the images containing the object (i.e. categories M, B, C and P) against non-object images from the database (i.e. category N). The training set contained 100 positive and 100 negative images. The tests are carried out on 100 new images, half belonging to the learned class and half not.

Like many other successful methods [3, 4] we take a "set-of-patches" approach to this problem. These methods represent an image in terms of the features of a set of small image patches. By carefully choosing the patches and their features this representation can be made largely robust to the common types of image transformation, e.g. scale, rotation, perspective, occlusion.

Two views were provided of each image through the use of different patch types. One was from affine invariant interest point detectors with a moment invariant descriptor calculated for each interest point. The second were key point features from SIFT detectors. For one image, several hundred characteristic patches were detected according to the complexity of the images. These were then clustered around $K = 400$ centres for each feature space. Each image is then represented as a histogram over these centres. So finally, for one image there are two feature vectors of length 400 that provide the two views.

|  | Motorbike | Bicycle | People | Car |
|---|---|---|---|---|
| SVM 1 | 94.05 | 91.58 | 91.58 | 87.95 |
| SVM 2 | 91.15 | 91.15 | 90.57 | 86.21 |
| KCCA + SVM | 94.19 | 90.28 | 90.57 | 88.68 |
| SVM 2K | **94.34** | **93.47** | **92.74** | **90.13** |

Table 1: Results for 4 datasets showing test accuracy of the individual SVMs and SVM-2K.

Figure 1 show the results of the test errors obtained for the different categories for the individual SVMs and the SVM-2K. There is a clear improvement in performance of the SVM-2K over the two individual SVMs in all four categories.

If we examine the structure of the optimisation, the restriction that the output of the two linear functions be similar seems to be an arbitrary restriction particularly for points that are far from the margin or are misclassified. Intuitively it would appear better to take advantage of the abilities of the different representations to better fit the data.

In order to understand this apparent contradiction we now consider a theoretical analysis of the generalisation of the SVM-2K using the framework provided by Rademacher complexity bounds.

## 4  Background theory

We begin with the definitions required for Rademacher complexity, see for example Bartlett and Mendelson [2] (see also [9] for an introductory exposition).

**Definition 1.** *For a sample* $S = \{\mathbf{x}_1, \cdots, \mathbf{x}_\ell\}$ *generated by a distribution* $D$ *on a set*

$X$ and a real-valued function class $\mathcal{F}$ with a domain $X$, the empirical Rademacher complexity of $\mathcal{F}$ is the random variable

$$\hat{R}_\ell\left(\mathcal{F}\right) = \mathbb{E}_\sigma\left[\sup_{f\in\mathcal{F}}\left|\frac{2}{\ell}\sum_{i=1}^{\ell}\sigma_i f\left(\mathbf{x}_i\right)\right|\,\middle|\,\mathbf{x}_1,\cdots,\mathbf{x}_\ell\right]$$

where $\sigma = \{\sigma_1,\cdots,\sigma_\ell\}$ are independent uniform $\{\pm 1\}$-valued Rademacher random variables. The Rademacher complexity of $\mathcal{F}$ is

$$R_\ell\left(\mathcal{F}\right) = \mathbb{E}_S\left[\hat{R}_\ell\left(\mathcal{F}\right)\right] = \mathbb{E}_{S\sigma}\left[\sup_{f\in\mathcal{F}}\left|\frac{2}{\ell}\sum_{i=1}^{\ell}\sigma_i f\left(\mathbf{x}_i\right)\right|\right]$$

We use $\mathbb{E}_\mathcal{D}$ to denote expectation with respect to a distribution $\mathcal{D}$ and $\mathbb{E}_S$ when the distribution is the uniform (empirical) distribution on a sample $S$.

**Theorem 1.** *Fix $\delta \in (0,1)$ and let $\mathcal{F}$ be a class of functions mapping from $S$ to $[0,1]$. Let $(\mathbf{x}_i)_{i=1}^{\ell}$ be drawn independently according to a probability distribution $\mathcal{D}$. Then with probability at least $1-\delta$ over random draws of samples of size $\ell$, every $f \in \mathcal{F}$ satisfies*

$$\mathbb{E}_\mathcal{D}\left[f\left(x\right)\right] \leq \mathbb{E}_S\left[f\left(x\right)\right] + R_\ell\left(\mathcal{F}\right) + 3\sqrt{\frac{\ln(2/\delta)}{2\ell}}$$
$$\leq \mathbb{E}_S\left[f\left(x\right)\right] + \hat{R}_\ell\left(\mathcal{F}\right) + 3\sqrt{\frac{\ln(2/\delta)}{2\ell}}$$

Given a training set $S$ the class of functions that we will primarily be considering are linear functions with bounded norm

$$\left\{x \to \sum_{i=1}^{\ell}\alpha_i\kappa\left(\mathbf{x}_i,x\right) : \alpha'K\alpha \leq B^2\right\}$$
$$\subseteq \left\{x \to \langle w,\phi\left(x\right)\rangle : \|w\| \leq B\right\} = \mathcal{F}_B$$

where $\phi$ is the feature mapping corresponding to the kernel $\kappa$ and $K$ is the corresponding kernel matrix for the sample $S$. The following result bounds the Rademacher complexity of linear function classes.

**Theorem 2.** *[2] If $\kappa : X \times X \to R$ is a kernel, and $S = \{\mathbf{x}_1,\cdots,\mathbf{x}_\ell\}$ is a sample of point from X, then the empirical Rademacher complexity of the class $\mathcal{F}_B$ satisfies*

$$\hat{R}_\ell\left(\mathcal{F}\right) \leq \frac{2B}{\ell}\sqrt{\sum_{i=1}^{\ell}\kappa\left(\mathbf{x}_i,\mathbf{x}_i\right)} = \frac{2B}{\ell}\sqrt{tr\left(K\right)}$$

### 4.1 Analysing SVM-2K

For SVM-2K, the two feature sets from the same objects are $(\phi_A\left(\mathbf{x}_i\right))_{i=1}^{\ell}$ and $(\phi_B\left(\mathbf{x}_i\right))_{i=1}^{\ell}$ respectively. We assume the notation and optimisation of SVM-2K given in section 2, equation (1).

First observe that an application of Theorem 1 shows that

$$\mathbb{E}_S[|f_A(x) - f_B(x)|] \leq \mathbb{E}_S[|\langle\hat{\mathbf{w}}_A,\phi_A(x)\rangle + \hat{b}_A - \langle\hat{\mathbf{w}}_B,\phi_B(x)\rangle - \hat{b}_B|]$$
$$\leq \epsilon + \frac{1}{\ell}\sum_{i=1}^{\ell}\eta_i + \frac{2C}{\ell}\sqrt{\operatorname{tr}(K_A) + \operatorname{tr}(K_B)} + 3\sqrt{\frac{\ln(2/\delta)}{2\ell}} =: D$$

with probability at least $1-\delta$. We have assumed that $\|\mathbf{w}_A\|^2 + b_A^2 \leq C^2$ and $\|\mathbf{w}_B\|^2 + b_B^2 \leq C^2$ for some prefixed $C$. Hence, the class of functions we are considering when applying

SVM-2K to this problem can be restricted to

$$\mathcal{F}_{C,D} = \left\{ f \middle| f : x \to 0.5 \left( \sum_{i=1}^{\ell} \left[ g_i^A \kappa_A \left( \mathbf{x}_i, x \right) + g_i^B \kappa_B \left( \mathbf{x}_i, x \right) \right] + b_A + b_B \right), \right.$$

$$\left. g^{A\prime} K_A g^A + b_A^2 \leq C^2, g^{B\prime} K_B g^B + b_B^2 \leq C^2, \mathbb{E}_S[|f_A(x) - f_B(x)|] \leq D \right\}$$

The class $\mathcal{F}_{C,D}$ is clearly closed under negation.

Applying the usual Rademacher techniques for margin bounds on generalisation we obtain the following result.

**Theorem 3.** *Fix* $\delta \in (0,1)$ *and let* $\mathcal{F}_{C,D}$ *be the class of functions described above. Let* $(\mathbf{x}_i)_{i=1}^{\ell}$ *be drawn independently according to a probability distribution* $\mathcal{D}$. *Then with probability at least* $1 - \delta$ *over random draws of samples of size* $\ell$, *every* $f \in \mathcal{F}_{C,D}$ *satisfies*

$$P_{(x,y)\sim\mathcal{D}}(\text{sign}(f(x)) \neq y) \leq \frac{0.5}{\ell} \sum_{i=1}^{\ell} (\xi_i^A + \xi_i^B) + \hat{R}_{\ell}(\mathcal{F}_{C,D}) + 3\sqrt{\frac{\ln(2/\delta)}{2\ell}}.$$

It therefore remains to compute the empirical Rademacher complexity of $\mathcal{F}_{C,D}$, which is the critical discriminator between the bounds for the individual SVMs and that of the SVM-2K.

## 4.2 Empirical Rademacher complexity of $\mathcal{F}_{C,D}$

We now define an auxiliary function of two weight vectors $\mathbf{w}_A$ and $\mathbf{w}_B$,

$$D(\mathbf{w}_A, \mathbf{w}_B) := \mathbb{E}_{\mathcal{D}}[|\langle \mathbf{w}_A, \phi_A(x) \rangle + b_A - \langle \mathbf{w}_B, \phi_B(x) \rangle - b_B|]$$

With this notation we can consider computing the Rademacher complexity of the class $\mathcal{F}_{C,D}$.

$$\hat{R}_{\ell}(\mathcal{F}_{C,D}) = \mathbb{E}_{\sigma} \left[ \sup_{f \in \mathcal{F}_{C,D}} \left| \frac{2}{\ell} \sum_{i=1}^{\ell} \sigma_i f(\mathbf{x}_i) \right| \right]$$

$$= \mathbb{E}_{\sigma} \left[ \sup_{\substack{\|\mathbf{w}_A\| \leq C, \ \|\mathbf{w}_B\| \leq C \\ D(\mathbf{w}_A, \mathbf{w}_B) \leq D}} \left| \frac{1}{\ell} \sum_{i=1}^{\ell} \sigma_i \left[ \langle \mathbf{w}_A, \phi_A(\mathbf{x}_i) \rangle + b_A + \langle \mathbf{w}_B, \phi_B(\mathbf{x}_i) \rangle + b_B \right] \right| \right]$$

Our next observation follows from a reversed version of the basic Rademacher complexity theorem reworked to reverse the roles of the empirical and true expectations:

**Theorem 4.** *Fix* $\delta \in (0,1)$ *and let* $\mathcal{F}$ *be a class of functions mapping from* $S$ *to* $[0,1]$. *Let* $(\mathbf{x}_i)_{i=1}^{\ell}$ *be drawn independently according to a probability distribution* $\mathcal{D}$. *Then with probability at least* $1 - \delta$ *over random draws of samples of size* $\ell$, *every* $f \in \mathcal{F}$ *satisfies*

$$\mathbb{E}_S[f(x)] \leq \mathbb{E}_{\mathcal{D}}[f(x)] + R_{\ell}(\mathcal{F}) + 3\sqrt{\frac{\ln(2/\delta)}{2\ell}}$$
$$\leq \mathbb{E}_{\mathcal{D}}[f(x)] + \hat{R}_{\ell}(\mathcal{F}) + 3\sqrt{\frac{\ln(2/\delta)}{2\ell}}$$

The proof tracks that of Theorem 1 but is omitted through lack of space.

For weight vectors $\mathbf{w}_A$ and $\mathbf{w}_B$ satisfying $D(\mathbf{w}_A, \mathbf{w}_B) \leq D$, an application of Theorem 4

shows that with probability at least $1 - \delta$ we have

$$
\begin{aligned}
\hat{D}(\mathbf{w}_A, \mathbf{w}_B) &:= \mathbb{E}_S[|\langle \mathbf{w}_A, \phi_A(x)\rangle + b_A - \langle \mathbf{w}_B, \phi_B(x)\rangle - b_B|] \\
&\leq D + \frac{2C}{\ell}\sqrt{\mathrm{tr}(K_A) + \mathrm{tr}(K_B)} + 3\sqrt{\frac{\ln(2/\delta)}{2\ell}} \\
&\leq \epsilon + \frac{1}{\ell}\sum_{i=1}^{\ell}\eta_i + \frac{4C}{\ell}\sqrt{\mathrm{tr}(K_A) + \mathrm{tr}(K_B)} + 6\sqrt{\frac{\ln(2/\delta)}{2\ell}} =: \hat{D}
\end{aligned}
$$

We now return to bounding the Rademacher complexity of $\mathcal{F}_{C,D}$. The above result shows that with probability greater than $1 - \delta$

$$
\begin{aligned}
&\hat{R}_\ell\big(\mathcal{F}_{C,D}\big) \\
&\leq \mathbb{E}_\sigma\left[ \sup_{\substack{\|\mathbf{w}_A\| \leq C \\ \|\mathbf{w}_B\| \leq C \\ \hat{D}(\mathbf{w}_A, \mathbf{w}_B) \leq \hat{D}}} \left|\tfrac{1}{\ell}\sum_{i=1}^{\ell}\sigma_i\left[\langle \mathbf{w}_A, \phi_A(\mathbf{x}_i)\rangle + b_A + \langle \mathbf{w}_B, \phi_B(\mathbf{x}_i)\rangle + b_B\right]\right| \right]
\end{aligned}
$$

First note that the expression in square brackets is concentrated under the uniform distribution of Rademacher variables. Hence, we can estimate the complexity for a fixed instantiation $\hat{\sigma}$ of the the Rademacher variables $\sigma$. We now must find the value of $\mathbf{w}_A$ and $\mathbf{w}_B$ that maximises the expression

$$
\begin{aligned}
&\frac{1}{\ell}\left|\left[\left\langle \mathbf{w}_A, \sum_{i=1}^{\ell}\hat{\sigma}_i\phi_A(\mathbf{x}_i)\right\rangle + b_A\sum_{i=1}^{\ell}\hat{\sigma}_i + \left\langle \mathbf{w}_B, \sum_{i=1}^{\ell}\hat{\sigma}_i\phi_B(\mathbf{x}_i)\right\rangle + b_B\sum_{i=1}^{\ell}\hat{\sigma}_i\right]\right| \\
&= \frac{1}{\ell}\left|\hat{\sigma}'K_A g^A + \hat{\sigma}'K_B g^B + (b_A + b_B)\hat{\sigma}'\mathbf{j}\right|
\end{aligned}
$$

subject to the constraints $g^{A\prime}K_A g^A \leq C^2$, $g^{B\prime}K_B g^B \leq C^2$, and

$$
\frac{1}{\ell}\mathbf{1}'\mathrm{abs}(K_A g^A - K_B g^B + (b_A - b_B)\mathbf{1}) \leq \hat{D}
$$

where $\mathbf{1}$ is the all ones vector and $\mathrm{abs}(\mathbf{u})$ is the vector obtained by applying the $\mathrm{abs}$ function to $\mathbf{u}$ component-wise. The resulting value of the objective function is the estimate of the Rademacher complexity. This is the optimisation solved in the brief experiments described below.

### 4.3 Experiments with Rademacher complexity

We computed the Rademacher complexity for the problems considered in the experimental section above. We wished to verify that the Rademacher complexity of the space $\mathcal{F}_{C,D}$, where $C$ and $D$ are determined by applying the SVM-2K, are indeed significantly lower than that obtained for the SVMs in each space individually.

|         | Motorbike | Bicycle | People | Car   |
|---------|-----------|---------|--------|-------|
| SVM 1   | 94.05     | 91.58   | 91.58  | 87.95 |
| Rad 1   | 1.65      | 0.93    | 0.91   | 1.60  |
| SVM 2   | 91.15     | 91.15   | 90.57  | 86.21 |
| Rad 2   | 1.72      | 1.48    | 0.87   | 1.64  |
| SVM 2K  | 94.34     | 93.47   | 92.74  | 90.13 |
| Rad 2K  | 1.26      | 1.28    | 0.82   | 1.26  |

Table 2: Results for 4 datasets showing test accuracy and Rademacher complexity (Rad) of the individual SVMs and SVM-2K.

Table 2 shows the results for the motorbike, bicycle, people and car datasets. We show the Rademacher complexities for the individual SVMs and for the SVM-2K along with the generalisation results already given in Table 1. In the case of SVM-2K we sampled the Rademacher variables 10 times and give the corresponding standard deviation. As predicted the Rademacher complexity is significantly smaller for SVM-2K, hence confirming the intuition that led to the introduction of the approach, namely that the complexity of the class is reduced by restricting the weight vectors to align on the training data. Provided both representations contain the necessary data we can therefore expect an improvement in generalisation as observed in the reported experiments.

## 5  Conclusions

With the plethora of data now being collected in a wide range of fields there is frequently the luxury of having two views of the same phenomenon. The simplest example is paired corpora of documents in different languages, but equally we can think of examples from bioinformatics, machine vision, etc. Frequently it is also reasonable to assume that both views contain all of the relevant information required for a classification task.

We have demonstrated that in such cases it can be possible to leaver the correlation between the two views to improve classification accuracy. This has been demonstrated in experiments with a machine vision task. Furthermore, we have undertaken a theoretical analysis to illuminate the source and extent of the advantage that can be obtained, showing in the cases considered a significant reduction in the Rademacher complexity of the corresponding function classes.

## Footnotes

[1]Available from `http://www.pascal-network.org/challenges/VOC/voc/160305_VOCdata.tar.gz`

## References

[1] Francis R. Bach and Michael I. Jordan. Kernel independent component analysis. *Journal of Machine Learning Research*, 3:1–48, 2002.

[2] P. L. Bartlett and S. Mendelson. Rademacher and Gaussian complexities: risk bounds and structural results. *Journal of Machine Learning Research*, 3:463–482, 2002.

[3] G. Csurka, C. Bray, C. Dance, and L. Fan. Visual categorization with bags of keypoints. In *XRCE Research Reports, XEROX*. The 8th European Conference on Computer Vision - ECCV, Prague, 2004.

[4] R. Fergus, P. Perona, and A. Zisserman. Object class recognition by unsupervised scale-invariant learning. In *Proceedings of the IEEE Conference on Computer Vision and Pattern Recognition*, 2003.

[5] David Hardoon, Sandor Szedmak, and John Shawe-Taylor. Canonical correlation analysis: An overview with application to learning methods. *Neural Computation*, 16:2639–2664, 2004.

[6] Yaoyong Li and John Shawe-Taylor. Using kcca for japanese-english cross-language information retrieval and classification. *to appear in Journal of Intelligent Information Systems*, 2005.

[7] S. Mika, B. Schölkopf, A. Smola, K.-R. Müller, M. Scholz, and G. Rätsch. Kernel PCA and de-noising in feature spaces. In *Advances in Neural Information Processing Systems 11*, 1998.

[8] R. Rosipal and L. J. Trejo. Kernel partial least squares regression in reproducing kernel hilbert space. *Journal of Machine Learning Research*, 2:97–123, 2001.

[9] J. Shawe-Taylor and N. Cristianini. *Kernel Methods for Pattern Analysis*. Cambridge University Press, Cambridge, UK, 2004.
